# Supervised Bipartite Graph Inference

**Yoshihiro Yamanishi**
Mines ParisTech  CBIO
Institut Curie, INSERM U900,
35 rue Saint-Honore, Fontainebleau, F-77300 France
`yoshihiro.yamanishi@ensmp.fr`

## Abstract

We formulate the problem of bipartite graph inference as a supervised learning problem, and propose a new method to solve it from the viewpoint of distance metric learning. The method involves the learning of two mappings of the heterogeneous objects to a unified Euclidean space representing the network topology of the bipartite graph, where the graph is easy to infer. The algorithm can be formulated as an optimization problem in a reproducing kernel Hilbert space. We report encouraging results on the problem of compound-protein interaction network reconstruction from chemical structure data and genomic sequence data.

## 1   Introduction

The problem of bipartite graph inference is to predict the presence or absence of edges between heterogeneous objects known to form the vertices of the bipartite graph, based on the observation about the heterogeneous objects. This problem is becoming a challenging issue in bioinformatics and computational biology, because there are many biological networks which are represented by a bipartite graph structure with vertices being heterogeneous molecules and edges being interactions between them. Examples include compound-protein interaction network consisting of interactions between ligand compounds and target proteins, metabolic network consisting of interactions between substrates and enzymes, and host-pathogen protein-protein network consisting of interactions between host proteins and pathogen proteins.

Especially, the prediction of compound-protein interaction networks is a key issue toward genomic drug discovery, because drug development depends heavily on the detection of interactions between ligand compounds and target proteins. The human genome sequencing project has made available the sequences of a large number of human proteins, while the high-throughput screening of large-scale chemical compound libraries is enabling us to explore the chemical space of possible compounds[1]. However, our knowledge about the such compound-protein interactions is very limited. It is therefore important is to detect unknown compound-protein interactions in order to identify potentially useful compounds such as imaging probes and drug leads from huge amount of chemical and genomic data.

A major traditional method for predicting compound-protein interactions is docking simulation [2]. However, docking simulation requires 3D structure information for the target proteins. Most pharmaceutically useful target proteins are membrane proteins such as ion channels and G protein-coupled receptors (GPCRs). It is still extremely difficult and expensive to determine the 3D structures of membrane proteins, which limits the use of docking. There is therefore a strong incentive to develop new useful prediction methods based on protein sequences, chemical compound structures, and the available known compound-protein interaction information simultaneously.

Recently, several supervised methods for inferring a simple graph structure (e.g., protein network, enzyme network) have been developed in the framework of kernel methods [3, 4, 5]. The corresponding algorithms of the previous methods are based on kernel canonical correlation analysis

[3], distance metric learning [4], and *em*-algorithm [5], respectively. However, the previous methods can only predict edges between homogeneous objects such as protein-protein interactions and enzyme-enzyme relations, so it is not possible to predict edges between heterogeneous objects such as compound-protein interactions and substrate-enzyme interactions, because their frameworks are based only on a simple graph structure with homogeneous vertices. In contrast, in this paper we address the problem of supervised learning of the bipartite graph rather than the simple graph.

In this contribution, we develop a new supervised method for inferring the bipartite graph, borrowing the idea of distance metric learning used in the framework for inferring the simple graph [4]. The proposed method involves the learning of two mappings of the heterogeneous objects to a unified Euclidean space representing the network topology of the bipartite graph, where the graph is easy to infer. The algorithm can be formulated as an optimization problem in a reproducing kernel Hilbert space. To our knowledge, there are no statistical methods to predict bipartite graphs from observed data in a supervised context. In the results, we show the usefulness of the proposed method on the predictions of compound-protein interaction network reconstruction from chemical structure data and genomic sequence data.

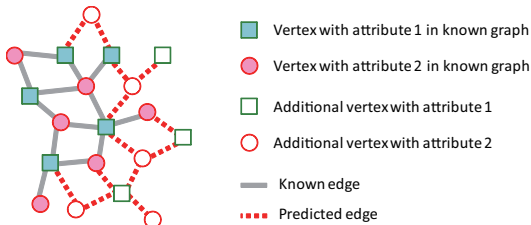

Figure 1: An illustration of the problem of the supervised bipartite graph inference

## 2   Formalism of the supervised bipartite graph inference problem

Let us formally define the supervised bipartite graph inference problem. Suppose that we are given an undirected bipartite graph $G = (U + V, E)$, where $U = (u_1, \cdots, u_{n_1})$ and $V = (v_1, \cdots, v_{n_2})$ are sets of heterogeneous vertices and $E \subset (U \times V) \cup (V \times U)$ is a set of edges. Note that the attribute of $U$ is completely different from that of $V$. The problem is, given additional sets of vertices $U' = (u'_1, \cdots, u'_{m_1})$ and $V' = (v'_1, \cdots, v'_{m_2})$, to infer a set of new edges $E' \subset U' \times (V + V') \cup V' \times (U + U') \cup (U + U') \times V' \cup (V + V') \times U'$ involving the additional vertices in $U'$ and $V'$. Figure 1 shows an illustration of this problem.

The prediction of compound-protein interaction networks is a typical problem which is suitable in this framework from a practical viewpoint. In this case, $U$ corresponds to a set of compounds (known ligands), $V$ corresponds to a set of proteins (known targets), and $E$ corresponds to a set of known compound-protein interactions (known ligand-target interactions). $U'$ corresponds to a set of additional compounds (new ligand candidates), $V'$ corresponds to a set of additional proteins (new target candidates), and $E'$ corresponds to a set of unknown compound-protein interactions (potential ligand-target interactions).

The prediction is performed based on available observations about the vertices. Sets of vertices $U = (u_1, \cdots, u_{n_1})$, $V = (v_1, \cdots, v_{n_2})$, $U' = (u'_1, \cdots, u'_{m_1})$ and $V' = (v'_1, \cdots, v'_{m_2})$ are represented by sets of observed data $\mathcal{X} = (x_1, \cdots, x_{n_1})$, $\mathcal{Y} = (y_1, \cdots, y_{n_2})$, $\mathcal{X}' = (x'_1, \cdots, x'_{m_1})$ and $\mathcal{Y}' = (y'_1, \cdots, y'_{m_2})$, respectively. For example, compounds are represented by molecular structures and proteins are represented by amino acid sequences. The question is how to predict unknown compound-protein interactions from compound structures and protein sequences using prior knowledge about known compound-protein interactions. Sets of $U$ and $V$ ($\mathcal{X}$ and $\mathcal{Y}$) are referred to as training sets, and heterogeneous objects are represented by $u$ and $v$ in the sense of vertices on the bipartite graph or by $x$ and $y$ in the sense of objects in the observed data below.

In order to deal with the data heterogeneity and take advantage of recent works on kernel similarity functions on general data structures [6], we will assume that $\mathcal{X}$ is a set endowed with a positive definite kernel $k_u$, that is, a symmetric function $k_u : \mathcal{X}^2 \to \mathbf{R}$ satisfying $\sum_{i,j=1}^{n_1} a_i a_j k_u(x_i, x_j) \geq 0$

for any $n_1 \in \mathbf{N}$, $(a_1, a_2, \cdots, a_{n_1}) \in \mathbf{R}^{n_1}$ and $(x_1, x_2, \cdots, x_{n_1}) \in \mathcal{X}$. Similarly, we will assume that $\mathcal{Y}$ is a set endowed with a positive definite kernel $k_v$, that is, a symmetric function $k_v : \mathcal{Y}^2 \to \mathbf{R}$ satisfying $\sum_{i,j=1}^{n_2} a_i a_j k_v(y_i, y_j) \geq 0$ for any $n_2 \in \mathbf{N}$, $(a_1, a_2, \cdots, a_{n_2}) \in \mathbf{R}^{n_2}$ and $(y_1, y_2, \cdots, y_{n_2}) \in \mathcal{Y}$.

## 3 Distance metric learning (DML) for the bipartite graph inference

### 3.1 Euclidean embedding and distance metric learning (DML)

Suppose that a bipartite graph must be reconstructed from the similarity information about $n_1$ objects $(x_1, \cdots, x_{n_1})$ in $\mathcal{X}$ (observed data for $U$) and $n_2$ objects $(y_1, \cdots, y_{n_2})$ in $\mathcal{Y}$ (observed data for $V$). One difficulty is that the attribute of observed data differs between $\mathcal{X}$ and $\mathcal{Y}$ in nature, so it is not possible to evaluate the link between $(x_1, \cdots, x_{n_1})$ and $(y_1, \cdots, y_{n_2})$ from the observed data directly. For example, in the case of compounds and proteins, each $x$ has a chemical graph structure and each $y$ has a sequence structure, so the data structures completely differ between $x$ and $y$. Therefore, we make an assumption that $n_1$ objects $(x_1, \cdots, x_{n_1})$ and $n_2$ objects $(y_1, \cdots, y_{n_2})$ are implicitly embedded in a unified Euclidean space $\mathbf{R}^d$, and a graph is inferred on those heterogeneous points by the nearest neighbor approach, i.e., putting an edge between heterogeneous points that are close to each other.

We propose the following two step procedure for the supervised bipartite graph inference:

1. embed the heterogeneous objects into a unified Euclidean space representing the network topology of the bipartite graph, where connecting heterogeneous vertices are close to each other, through mappings $f : \mathcal{X} \to \mathbf{R}^d$ and $g : \mathcal{Y} \to \mathbf{R}^d$

2. apply the mappings $f$ and $g$ to $\mathcal{X}'$ and $\mathcal{Y}'$ respectively, and predict new edges between the heterogeneous objects if the distance between the points $\{f(x), x \in \mathcal{X} \cup \mathcal{X}'\}$ and $\{g(y), y \in \mathcal{Y} \cup \mathcal{Y}'\}$ is smaller than a fixed threshold $\delta$.

While the second step in this procedure is fixed, the first step can be optimized by supervised learning of $f$ and $g$ using the known bipartite graph. To do so, we require the mappings $f$ and $g$ to map adjacent heterogeneous vertices in the known bipartite graph onto nearby positions in a unified Euclidian space $\mathbf{R}^d$, in order to ensure that the known bipartite graph can be recovered to some extent by the nearest neighbor approach.

Given functions $f : \mathcal{X} \to \mathbf{R}$ and $g : \mathcal{Y} \to \mathbf{R}$, a possible criterion to assess whether connected (resp. disconnected) heterogeneous vertices are mapped onto similar (resp. dissimilar) points in $\mathbf{R}$ is the following:

$$R(f,g) = \frac{\sum_{(u_i,v_j)\in E}(f(x_i) - g(y_j))^2 - \sum_{(u_i,v_j)\notin E}(f(x_i) - g(y_j))^2}{\sum_{(u_i,v_j)\in U\times V}(f(x_i) - g(y_j))^2}. \tag{1}$$

A small value of $R(f,g)$ ensures that connected heterogeneous vertices tend to be closer than disconnected heterogeneous vertices in the sense of quadratic error.

To represent the connectivity between heterogeneous vertices on the bipartite graph $G = (U+V, E)$, we define a kind of the adjacency matrix $A_{uv}$, where element $(A_{uv})_{ij}$ is equal to 1 (resp. 0) if vertices $u_i$ and $v_j$ are connected (resp. disconnected). Note that the size of the matrix $A_{uv}$ is $n_1 \times n_2$. We also define a kind of the degree matrix of the heterogeneous vertices as $D_u$ and $D_v$, where diagonal elements $(D_u)_{ii}$ and $(D_v)_{jj}$ are the degrees of vertices $u_i$ and $v_j$ (the numbers of edges involving vertices $u_i$ and $v_j$), respectively. Note that all non-diagonal elements in $D_u$ and $D_v$ are zero, and the sizes of the matrices are $n_1 \times n_1$ and $n_2 \times n_2$, respectively.

Let us denote by $f_U = (f(x_1), \cdots, f(x_{n_1}))^T \in \mathbf{R}^{n_1}$ and $g_V = (g(y_1), \cdots, g(y_{n_2}))^T \in \mathbf{R}^{n_2}$ the values taken by $f$ and $g$ on the training set. If we restrict $f_U$ and $f_V$ to have zero means as $\sum_{i=1}^{n_1} f(x_i) = 0$ and $\sum_{i=1}^{n_2} g(y_i) = 0$, then the criterion (1) can be rewritten as follows:

$$R(f,g) = 4 \frac{\begin{pmatrix} f_U \\ g_V \end{pmatrix}^T \begin{pmatrix} D_u & -A_{uv} \\ -A_{uv}^T & D_v \end{pmatrix} \begin{pmatrix} f_U \\ g_V \end{pmatrix}}{\begin{pmatrix} f_U \\ g_V \end{pmatrix}^T \begin{pmatrix} f_U \\ g_V \end{pmatrix}} - 2 \tag{2}$$

To avoid the over-fitting problem and obtain meaningful solutions, we propose to regularize the criterion (1) by a smoothness functional on $f$ and $g$ based on a classical approach in statistical learning [7, 8]. We assume that that $f$ and $g$ belong to the reproducing kernel Hilbert space (r.k.h.s.) $\mathcal{H}_U$ and $\mathcal{H}_V$ defined by the kernels $k_u$ on $\mathcal{X}$ and $k_v$ on $\mathcal{Y}$, and to use the norms of $f$ and $g$ as regularization operators. Let us define by $||f||$ and $||g||$ the norms of $f$ and $g$ in $\mathcal{H}_U$ and $\mathcal{H}_V$. Then, the regularized criterion to be minimized becomes:

$$R(f,g) = \frac{\begin{pmatrix} f_U \\ g_V \end{pmatrix}^T \begin{pmatrix} D_u & -A_{uv} \\ -A_{uv}^T & D_v \end{pmatrix} \begin{pmatrix} f_U \\ g_V \end{pmatrix} + \lambda_1||f||^2 + \lambda_2||g||^2}{\begin{pmatrix} f_U \\ g_V \end{pmatrix}^T \begin{pmatrix} f_U \\ g_V \end{pmatrix}}, \qquad (3)$$

where $\lambda_1$ and $\lambda_2$ are regularization parameters which control the trade-off between minimizing the original criterion (1) and ensuring that the solution has a small norm in the r.k.h.s.

The criterion is defined up to a scaling of the functions and the solution is therefore a direction in the r.k.h.s. Here we set additional constraints. In this case we impose the norm $||f|| = ||g|| = 1$, which corresponds to an orthogonal projection onto the direction selected in the r.k.h.s. Note that the criterion can be used for extracting one-dimentional feature of the objects. In order to obtain a $d$-dimensional feature representation of the objects, we propose to iterate the minimization of the regularized criterion (3) under orthogonality constraints in the r.k.h.s., that is, we recursively define the $p$-th features $f_p$ and $g_p$ for $p = 1, \cdots, d$ as follows:

$$(f_p, g_p) = \arg\min \frac{\begin{pmatrix} f_U \\ g_V \end{pmatrix}^T \begin{pmatrix} D_u & -A_{uv} \\ -A_{uv}^T & D_v \end{pmatrix} \begin{pmatrix} f_U \\ g_V \end{pmatrix} + \lambda_1||f||^2 + \lambda_2||g||^2}{\begin{pmatrix} f_U \\ g_V \end{pmatrix}^T \begin{pmatrix} f_U \\ g_V \end{pmatrix}} \qquad (4)$$

under the orthogonality constraints: $f \perp f_1, \cdots, f_{p-1}$, and $g \perp g_1, \cdots, g_{p-1}$.

In the prediction process, we map any new objects $x' \in \mathcal{X}'$ and $y' \in \mathcal{Y}'$ by the mappings $f$ and $g$ respectively, and predict new edges between the heterogeneous objects if the distance between the points $\{f(x), x \in \mathcal{X} \cup \mathcal{X}'\}$ and $\{g(y), y \in \mathcal{Y} \cup \mathcal{Y}'\}$ is smaller than a fixed threshold $\delta$.

## 3.2 Algorithm

Let $k_u$ and $k_v$ be the kernels on the sets $\mathcal{X}$ and $\mathcal{Y}$, where the kernels are both centered in $\mathcal{H}_U$ and $\mathcal{H}_V$. According to the representer theorem [9] in the r.k.h.s., for any $p = 1, \cdots, d$, the solution to equation (4) has the following expansions:

$$f_p(x) = \sum_{j=1}^{n_1} \alpha_{p,j} k_u(x_j, x), \quad g_p(y) = \sum_{j=1}^{n_2} \beta_{p,j} k_v(y_j, y), \qquad (5)$$

for some vector $\boldsymbol{\alpha}_p = (\alpha_{p,1}, \cdots, \alpha_{p,n_1})^T \in \mathbf{R}^{n_1}$ and $\boldsymbol{\beta}_p = (\beta_{p,1}, \cdots, \beta_{p,n_2})^T \in \mathbf{R}^{n_2}$.

Let $K_u$ and $K_v$ be the Gram matrices of the kernels $k_u$ and $k_u$ such that $(K_u)_{ij} = k_u(x_i, x_j)$, $i, j = 1, \cdots, n_1$ and $(K_v)_{ij} = k_v(y_i, y_j)$, $i, j = 1, \cdots, n_2$. The corresponding feature vectors $f_{p,U}$ and $g_{p,V}$ can be written as $f_{p,U} = K_u \boldsymbol{\alpha}_p$ and $g_{p,V} = K_v \boldsymbol{\beta}_p$, respectively. The squared norms of features $f$ and $g$ in $\mathcal{H}_U$ and $\mathcal{H}_V$ are equal to $||f||^2 = \boldsymbol{\alpha}^T K_u \boldsymbol{\alpha}$ and $||g||^2 = \boldsymbol{\beta}^T K_v \boldsymbol{\beta}$, so the normalization constraints for $f$ and $g$ can be written as $\boldsymbol{\alpha}^T K_u \boldsymbol{\alpha} = \boldsymbol{\beta}^T K_v \boldsymbol{\beta} = 1$. The orthogonarity constraints $f_p \perp f_q$ and $g_p \perp g_q$ $(p \neq q)$ can be written by $\boldsymbol{\alpha}_p^T K_u \boldsymbol{\alpha}_q = 0$ and $\boldsymbol{\beta}_p^T K_v \boldsymbol{\beta}_q = 0$.

Using the above representations, the minimization problem of $R(f,g)$ is equivalent to finding $\boldsymbol{\alpha}$ and $\boldsymbol{\beta}$ which minimize

$$R(f,g) = \frac{\begin{pmatrix} \boldsymbol{\alpha} \\ \boldsymbol{\beta} \end{pmatrix}^T \begin{pmatrix} K_u D_u K_u & -K_u A_{uv} K_v \\ -K_v A_{uv}^T K_u & K_v D_v K_v \end{pmatrix} \begin{pmatrix} \boldsymbol{\alpha} \\ \boldsymbol{\beta} \end{pmatrix} + \lambda_1 \boldsymbol{\alpha}^T K_u \boldsymbol{\alpha} + \lambda_2 \boldsymbol{\beta}^T K_v \boldsymbol{\beta}}{\begin{pmatrix} \boldsymbol{\alpha} \\ \boldsymbol{\beta} \end{pmatrix}^T \begin{pmatrix} K_u K_u & 0 \\ 0 & K_v K_v \end{pmatrix} \begin{pmatrix} \boldsymbol{\alpha} \\ \boldsymbol{\beta} \end{pmatrix}}, \qquad (6)$$

under the following orthogonality constraints:

$$\boldsymbol{\alpha}^T K_u \boldsymbol{\alpha}_1 = \cdots = \boldsymbol{\alpha}^T K_u \boldsymbol{\alpha}_{(p-1)} = 0, \quad \boldsymbol{\beta}^T K_v \boldsymbol{\beta}_1 = \cdots = \boldsymbol{\beta}^T K_v \boldsymbol{\beta}_{(p-1)} = 0.$$

Taking the differential of equation (6) with respect to $\boldsymbol{\alpha}$ and $\boldsymbol{\beta}$ and setting to zero, the solution of the first vectors $\boldsymbol{\alpha}_1$ and $\boldsymbol{\beta}_1$ can be obtained as the eigenvectors associated with the smallest (non-negative) eigenvalue in the following generalized eigenvalue problem:

$$\begin{pmatrix} K_u D_u K_u + \lambda_1 K_u & -K_u A_{uv} K_v \\ -K_v A_{uv}^T K_u & K_v D_v K_v + \lambda_2 K_v \end{pmatrix} \begin{pmatrix} \boldsymbol{\alpha} \\ \boldsymbol{\beta} \end{pmatrix} = \rho \begin{pmatrix} K_u K_u & 0 \\ 0 & K_v K_v \end{pmatrix} \begin{pmatrix} \boldsymbol{\alpha} \\ \boldsymbol{\beta} \end{pmatrix} \quad (7)$$

Sequentially, the solutions of vectors $\boldsymbol{\alpha}_1, \cdots, \boldsymbol{\alpha}_d$ and $\boldsymbol{\beta}_1, \cdots, \boldsymbol{\beta}_d$ can be obtained as the eigenvectors associated with $d$ smallest (non-negative) eigenvalues in the above generalized eigenvalue problem.

## 4   Relationship with other methods

The process of embedding heterogeneous objects into the same space is similar to correspondence analysis (CA) [10] and Co-Occurence Data Embedding (CODE) [11] which are unsupervised methods to embed the rows and columns of a contingency table (adjacency matrix $A_{uv}$ in this study) into a low dimensional Euclidean space. However, critical differences with our proposed method are as follows: i) the above methods cannot use observed data ($\mathcal{X}$ and $\mathcal{Y}$ in this study) about heterogeneous nodes for prediction, because the algorithms are based only on co-occurence information ($A_{uv}$ in this study), and ii) we need to define a new representation of not only the objects in the training set but also additional objects outside of the training set. Therefore, it is not possible to directly apply the above methods to the bipartite graph inference problem.

Recall that the goal of the ordinary CA is to find embedding functions $\phi : U \to \mathbf{R}$ and $\psi : V \to \mathbf{R}$ which maximize the following correlation coefficient:

$$corr(\phi, \psi) = \frac{\sum_{i,j} I\{(u_i, v_j) \in E\} \phi(u_i) \psi(v_j)}{\sqrt{\sum_i d_{u_i} \phi(u_i)^2} \sqrt{\sum_j d_{v_j} \psi(v_j)^2}}, \quad (8)$$

where $I\{\cdot\}$ is an indicator function which returns 1 if the argument is true or 0 otherwise, $d_{u_i}$ (resp. $d_{v_j}$) is the degree of node $u_i$ (resp. $v_j$), and $\sum_i \phi(u_i) = 0$ (resp. $\sum_j \psi(v_j) = 0$) is assumed [10]. Here we attempt to consider an extension of the CA using the idea of kernel methods so that it can work in the context of the bipartite graph inference problem. The method is referred to as kernel correspondence analysis (KCA) below.

To formulate the KCA, we propose to replace the embedding functions $\phi : U \to \mathbf{R}$ and $\psi : V \to \mathbf{R}$ by functions $f : \mathcal{X} \to \mathbf{R}$ and $g : \mathcal{Y} \to \mathbf{R}$, where $f$ and $g$ belong to the r.k.h.s. $\mathcal{H}_U$ and $\mathcal{H}_V$ defined by the kernels $k_u$ on $\mathcal{X}$ and $k_v$ on $\mathcal{Y}$. Then, we consider maximizing the following regularized correlation coefficient:

$$corr(f, g) = \frac{\sum_{i,j} I\{(u_i, v_j) \in E\} f(x_i) g(y_j)}{\sqrt{\sum_i d_{u_i} f(x_i)^2 + \lambda_1 ||f||^2} \sqrt{\sum_j d_{v_j} g(y_j)^2 + \lambda_2 ||g||^2}}, \quad (9)$$

where $\lambda_1$ and $\lambda_2$ are regularization parameters which control the trade-off between maximizing the original correlation coefficient between two features and ensuring that the solution has a small norm in the r.k.h.s. In order to obtain a $d$-dimensional feature representation and deal with the scale issue, we propose to iterate the maximization of the regularized correlation coefficient (9) under orthogonality constraints in the r.k.h.s., that is, we recursively define the $p$-th features $f_p$ and $g_p$ for $p = 1, \cdots, d$ as $(f_p, g_p) = \arg\max corr(f, g)$ under the orthogonality constraints: $f \perp f_1, \cdots, f_{p-1}$ and $g \perp g_1, \cdots, g_{p-1}$ and the normalization constraints: $||f|| = ||g|| = 1$.

Using the function expansions in equation (5) and related matrix representations defined in the previous section, the maximization problem of the regularized correlation coefficient in equation (9) is equivalent to finding $\boldsymbol{\alpha}$ and $\boldsymbol{\beta}$ which maximize

$$corr(f, g) = \frac{\boldsymbol{\alpha}^T K_u A_{uv} K_v \boldsymbol{\beta}}{\sqrt{\boldsymbol{\alpha}^T K_u D_u K_u \boldsymbol{\alpha} + \lambda_1 \boldsymbol{\alpha}^T K_u \boldsymbol{\alpha}} \sqrt{\boldsymbol{\beta}^T K_v D_v K_v \boldsymbol{\beta} + \lambda_2 \boldsymbol{\beta}^T K_v \boldsymbol{\beta}}}. \quad (10)$$

Taking the differential of equation (10) with respect to $\boldsymbol{\alpha}$ and $\boldsymbol{\beta}$ and setting to zero, the solution of the first vectors $\boldsymbol{\alpha}_1$ and $\boldsymbol{\beta}_1$ can be obtained as the eigenvectors associated with the largest eigenvalue in the following generalized eigenvalue problem:

$$\begin{pmatrix} 0 & K_u A_{uv} K_v \\ K_v A_{uv}^T K_u & 0 \end{pmatrix} \begin{pmatrix} \boldsymbol{\alpha} \\ \boldsymbol{\beta} \end{pmatrix} = \rho \begin{pmatrix} K_u D_u K_u + \lambda_1 K_u & 0 \\ 0 & K_v D_v K_v + \lambda_2 K_v \end{pmatrix} \begin{pmatrix} \boldsymbol{\alpha} \\ \boldsymbol{\beta} \end{pmatrix}.$$
(11)

Sequentially, the solutions of vectors $\boldsymbol{\alpha}_1, \cdots, \boldsymbol{\alpha}_d$ and $\boldsymbol{\beta}_1, \cdots, \boldsymbol{\beta}_d$ can be obtained as the eigenvectors associated with $d$ largest eigenvalues in the above generalized eigenvalue problem.

The final form of KCA is similar to that of kernel canonical correlation analysis (KCCA) [12, 13], so KCA can be regarded as a variant of KCCA. However, the critical differences between KCA and KCCA are as follows: i) the objects are the same across two different data in KCCA, while the objects are different across two different data in KCA, and ii) KCCA cannot deal with co-occurence information about the objects. In the experiment below, we are interested in the performance comparison between the distance learning in DML and correlation maximization in KCA. A similar extension might be possible for CODE as well, but it is out of scope in this paper.

## 5 Experiment

### 5.1 Data

In this study we focus on compound-protein interaction networks made by four pharmaceutically useful protein classes: enzymes, ion channels, G protein-coupled receptors (GPCRs), and nuclear receptors. The information about compound-protein interactions were obtained from the KEGG BRITE [14], SuperTarget [15] and DrugBank databases [16]. The number of known interactions involving enzymes, ion channels, GPCRs, and nuclear receptors is 5449, 3020, 1124, and 199, respectively. The number of proteins involving the interactions is 1062, 242, 134, and 38, respectively, and the number of compounds involving the interactions is 1123, 475, 417, and 115, respectively. The compound set includes not only drugs but also experimentally confirmed ligand compounds. These data are regarded as gold standard sets to evaluate the prediction performance below.

Chemical structures of the compounds and amino acid sequences of the human proteins were obtained from the KEGG database [14]. We computed the kernel similarity value of chemical structures between compounds using the SIMCOMP algorithm [17], where the kernel similarity value between two compounds is computed by Tanimoto coefficient defined as the ratio of common substructures between two compounds based on a graph alignment algorithm. We computed the sequence similarities between the proteins using Smith-Waterman scores based on the local alignment between two amino acid sequences [18]. In this study we used the above similarity measures as kernel functions, but the Smith-Waterman scores are not always positive definite, so we added an appropriate identify matrix such that the corresponding kernel Gram matrix is positive definite, which is related with [19]. All the kernel matrices are normalized such that all diagonals are ones.

### 5.2 Performance evaluation

As a baseline method, we used the nearest neighbor (NN) method, because this idea has been used in traditional molecular screening in many public databases. Given a new ligand candidate compound, we find a known ligand compound (in the training set) sharing the highest structure similarity with the new compound, and predict the new compound to interact with proteins known to interact with the nearest ligand compound. Likewise, given a new target candidate protein, we find a known target protein (in the training set) sharing the highest sequence similarity with the new protein, and predict the new protein to interact with ligand compounds known to interact with the nearest target protein. Newly predicted compound-protein interaction pairs are assigned prediction scores with the highest structure or sequence similarity values involving new compounds or new proteins in order to draw the ROC curve below.

We tested the three different methods: NN, KCA, and DML on their abilities to reconstruct the four compound-protein interaction networks. We performed the following 5-fold cross-validation procedure: the gold standard set was split into 5 subsets of roughly equal size by compounds and proteins, each subset was then taken in turn as a test set, and the training is performed on the remaining 4

Table 1: AUC (ROC scores) for each interaction class, where "train c.", "train p.", "test c.", and "test p." indicates training compounds, training proteins, test compounds and test proteins, respectively.

| Data | Prediction class | Nearest neighbor (NN) | Kernel correspondence analysis (KCA) | Distance metric learning (DML) |
|---|---|---|---|---|
| Enzyme | i) test c. vs train p. | $0.655 \pm 0.011$ | $0.741 \pm 0.011$ | $0.843 \pm 0.006$ |
| | ii) train c. vs test p. | $0.758 \pm 0.008$ | $0.839 \pm 0.009$ | $0.878 \pm 0.003$ |
| | iii) test c. vs test p. | $0.500 \pm 0.000$ | $0.692 \pm 0.008$ | $0.782 \pm 0.013$ |
| | iv) all c. vs all p. | $0.684 \pm 0.006$ | $0.778 \pm 0.008$ | $0.852 \pm 0.020$ |
| Ion channel | i) test c. vs train p. | $0.712 \pm 0.004$ | $0.768 \pm 0.008$ | $0.800 \pm 0.004$ |
| | ii) train c. vs test p. | $0.896 \pm 0.008$ | $0.927 \pm 0.004$ | $0.945 \pm 0.002$ |
| | iii) test c. vs test p. | $0.500 \pm 0.000$ | $0.748 \pm 0.004$ | $0.771 \pm 0.008$ |
| | iv) all c. vs all p. | $0.770 \pm 0.004$ | $0.838 \pm 0.005$ | $0.864 \pm 0.002$ |
| GPCR | i) test c. vs train p. | $0.714 \pm 0.005$ | $0.848 \pm 0.002$ | $0.882 \pm 0.005$ |
| | ii) train c. vs test p. | $0.781 \pm 0.026$ | $0.895 \pm 0.025$ | $0.936 \pm 0.004$ |
| | iii) test c. vs test p. | $0.500 \pm 0.000$ | $0.823 \pm 0.038$ | $0.864 \pm 0.013$ |
| | iv) all c. vs all p. | $0.720 \pm 0.013$ | $0.866 \pm 0.015$ | $0.904 \pm 0.003$ |
| Nuclear receptor | i) test c. vs train p. | $0.715 \pm 0.009$ | $0.808 \pm 0.018$ | $0.832 \pm 0.013$ |
| | ii) train c. vs test p. | $0.683 \pm 0.010$ | $0.784 \pm 0.012$ | $0.812 \pm 0.036$ |
| | iii) test c. vs test p. | $0.500 \pm 0.000$ | $0.670 \pm 0.053$ | $0.747 \pm 0.049$ |
| | iv) all c. vs all p. | $0.675 \pm 0.004$ | $0.784 \pm 0.011$ | $0.815 \pm 0.024$ |

sets. We draw a receiver operating curve (ROC), the plot of true positives as a function of false positives based on various thresholds $\delta$, where true positives are correctly predicted interactions and false positives are predicted interactions that are not present in the gold standard interactions. The performance was evaluated by AUC (area under the ROC curve) score. The regularization parameter $\lambda$ and the number of features $d$ are optimized by applying the internal cross-validation within the training set with the AUC score as a target criterion in the case of KCA and DML. To obtain robust results, we repeated the above cross-validation experiment five times, and computed the average and standard deviation of the resulting AUC scores.

Table 1 shows the resulting AUC scores for different sets of predictions depending on whether the compound and/or the protein were in the initial training set or not. Compounds and proteins in the training set are called training compounds and proteins whereas those not in the training set are called test compounds and proteins. Four different classes are then possible: i) test compounds vs training proteins, ii) training compounds vs test proteins, iii) test compounds vs test proteins, and iv) all the possible predictions (the average of the above three parts). Comparing the three different methods, DML seems to have the best performance for all four types of compound-protein interaction networks, and outperform the other methods KCA and NN at a significant level. The worst performance of NN implies that raw compound structure or protein sequence similarities do not always reflect the tendency of interaction partners in true compound-protein interaction networks. Among the four prediction classes, predictions where neither the protein nor the compound are in the training set (iii) are weakest, but even then reliable predictions are possible in DML. Note that the NN method cannot predict iii) test vs test interaction, because it depends on the template information about known ligand compounds and known target proteins. These results suggest that the feature space learned by DML successfully represents the network topology of the bipartite graph structure of compound-protein networks, and the correlation maximization learning used in KCA is not enough to reflect the network topology of the bipartite graph.

## 6 Concluding remarks

In this paper, we developed a new supervised method to infer the bipartite graph from the viewpoint of distance metric learning (DML). The originality of the proposed method lies in the embedding of heterogeneous objects forming vertices on the bipartite graph into a unified Euclidian space and in the learning of the distance between heterogeneous objects with different data structures in the unified feature space. We also discussed the relationship with correspondence analysis (CA) and kernel canonical correlation analysis (KCCA). In the experiment, it is shown that the proposed method DML outperforms the other methods on the problem of compound-protein interaction network reconstruction from chemical structure and genomic sequence data. From a practical viewpoint, the

proposed method is useful for virtual screening of a huge number of ligand candidate compounds being generated with various biological assays and target candidate proteins toward genomic drug discovery. It should be also pointed out that the proposed method can be applied to other network prediction problems such as metabolic network reconstruction, host-pathogen protein-protein interaction prediction, and customer-product recommendation system as soon as they are represented by bipartite graphs.

## References

[1] C.M. Dobson. Chemical space and biology. *Nature*, 432:824–828, 2004.

[2] M. Rarey, B. Kramer, T. Lengauer, and G. Klebe. A fast flexible docking method using an incremental construction algorithm. *J Mol Biol*, 261:470–489, 1996.

[3] Y. Yamanishi, J.P. Vert, and M. Kanehisa. Protein network inference from multiple genomic data: a supervised approach. *Bioinformatics*, 20 Suppl 1:i363–370, 2004.

[4] J.-P. Vert and Y. Yamanishi. Supervised graph inference. *Advances in Neural Information and Processing System*, pages 1433–1440, 2005.

[5] T. Kato, K. Tsuda, and K. Asai. Selective integration of multiple biological data for supervised network inference. *Bioinformatics*, 21:2488–2495, 2005.

[6] B. Schölkopf, K. Tsuda, and J.P. Vert. *Kernel Methods in Computational Biology*. MIT Press, 2004.

[7] G. Wahba. *Splines Models for Observational Data: Series in Applied Mathematics*. SIAM, Philadelphia, 1990.

[8] F. Girosi, M. Jones, and T. Poggio. Regularization theory and neural networks architectures. *Neural Computation*, 7:219–269, 1995.

[9] J. Shawe-Taylor and N. Cristianini. *Kernel Methods for Pattern Analysis*. Camb. Univ. Press, 2004.

[10] M.J. Greenacre. *Theory and applications of correspondence analysis*. Academic Press, 1984.

[11] A. Globerson, G. Chechik, F. Pereira, and N. Tishby. Euclidean embedding of co-occurrence data. *Advances in Neural Information and Processing System*, pages 497–504, 2005.

[12] S. Akaho. A kernel method for canonical correlation analysis. *International. Meeting on Psychometric Society (IMPS2001)*, 2001.

[13] F.R. Bach and M.I. Jordan. Kernel independent component analysis. *Journal of Machine Learning Research*, 3:1–48, 2002.

[14] M. Kanehisa, S. Goto, M. Hattori, K.F. Aoki-Kinoshita, M. Itoh, S. Kawashima, T. Katayama, M. Araki, and M. Hirakawa. From genomics to chemical genomics: new developments in kegg. *Nucleic Acids Res.*, 34(Database issue):D354–357, Jan 2006.

[15] S. Gunther, S. Guenther, M. Kuhn, M. Dunkel, and et al. Supertarget and matador: resources for exploring drug-target relationships. *Nucleic Acids Res*, 2007.

[16] D.S. Wishart, C. Knox, A.C. Guo, D. Cheng, S. Shrivastava, D. Tzur, B. Gautam, and M. Hassanali. Drugbank: a knowledgebase for drugs, drug actions and drug targets. *Nucleic Acids Res*, 2007.

[17] M. Hattori, Y. Okuno, S. Goto, and M. Kanehisa. Development of a chemical structure comparison method for integrated analysis of chemical and genomic information in the metabolic pathways. *J. Am. Chem. Soc.*, 125:11853–11865, 2003.

[18] T.F. Smith and M.S. Waterman. Identification of common molecular subsequences. *J Mol Biol*, 147:195–197, 1981.

[19] H. Saigo, J.P. Vert, N. Ueda, and T. Akutsu. Protein homology detection using string alignment kernels. *Bioinformatics*, 20:1682–1689, 2004.

